# Learning winner-take-all competition between groups of neurons in lateral inhibitory networks

**Xiaohui Xie, Richard Hahnloser and H. Sebastian Seung**
E25-210, MIT, Cambridge, MA 02139
{xhxie|rh|seung}@mit.edu

## Abstract

It has long been known that lateral inhibition in neural networks can lead to a winner-take-all competition, so that only a single neuron is active at a steady state. Here we show how to organize lateral inhibition so that groups of neurons compete to be active. Given a collection of potentially overlapping groups, the inhibitory connectivity is set by a formula that can be interpreted as arising from a simple learning rule. Our analysis demonstrates that such inhibition generally results in winner-take-all competition between the given groups, with the exception of some degenerate cases. In a broader context, the network serves as a particular illustration of the general distinction between permitted and forbidden sets, which was introduced recently. From this viewpoint, the computational function of our network is to store and retrieve memories as permitted sets of coactive neurons.

In traditional winner-take-all networks, lateral inhibition is used to enforce a localized, or "grandmother cell" representation in which only a single neuron is active [1, 2, 3, 4]. When used for unsupervised learning, winner-take-all networks discover representations similar to those learned by vector quantization [5]. Recently many research efforts have focused on unsupervised learning algorithms for sparsely distributed representations [6, 7]. These algorithms lead to networks in which groups of multiple neurons are coactivated to represent an object. Therefore, it is of great interest to find ways of using lateral inhibition to mediate winner-take-all competition between groups of neurons, as this could be useful for learning sparsely distributed representations.

In this paper, we show how winner-take-all competition between groups of neurons can be learned. Given a collection of potentially overlapping groups, the inhibitory connectivity is set by a simple formula that can be interpreted as arising from an online learning rule. To show that the resulting network functions as advertised, we perform a stability analysis. If the strength of inhibition is sufficiently great, and the group organization satisfies certain conditions, we show that the only sets of neurons that can be coactivated at a stable steady state are the given groups and their subsets. Because of the competition between groups, only one group can be activated at a time. In general, the identity of the winning group depends on the initial conditions of the network dynamics. If the groups are ordered by the aggregate input that each receives, the possible winners are those above a cutoff that is set by inequalities to be specified.

# 1 Basic definitions

Let $m$ groups of neurons be given, where group membership is specified by the matrix

$$\xi_i^a = \begin{cases} 1 & \text{if the } i\text{th neuron is in the } a\text{th group} \\ 0 & \text{otherwise} \end{cases} \tag{1}$$

We will assume that every neuron belongs to at least one group[1], and every group contains at least one neuron. A neuron is allowed to belong to more than one group, so that the groups are potentially overlapping. The inhibitory synaptic connectivity of the network is defined in terms of the group membership,

$$J_{ij} = \prod_{a=1}^{m}(1 - \xi_i^a \xi_j^a) = \begin{cases} 0 & \text{if } i \text{ and } j \text{ both belong to a group} \\ 1 & \text{otherwise} \end{cases} \tag{2}$$

One can imagine this pattern of connectivity arising by a simple learning mechanism. Suppose that all elements of $J$ are initialized to be unity, and the groups are presented sequentially as binary vectors $\xi^1, \dots, \xi^m$. The $a$th group is learned through the update

$$J_{ij} \leftarrow J_{ij}(1 - \xi_i^a \xi_j^a) \tag{3}$$

In other words, if neurons $i$ and $j$ both belong to group $a$, then the connection between them is removed. After presentation of all $m$ groups, this leads to Eq. (2). At the start of the learning process, the initial state of $J$ corresponds to uniform inhibition, which is known to implement winner-take-all competition between individual neurons. It will be seen that, as inhibitory connections are removed during learning, the competition evolves to mediate competition between groups of neurons rather than individual neurons.

The dynamics of the network is given by

$$\frac{dx_i}{dt} + x_i = \left[ b_i + \alpha x_i - \beta \sum_j J_{ij} x_j \right]^+ \tag{4}$$

where $[z]^+ = \max\{z, 0\}$ denotes rectification, $\alpha > 0$ the strength of self-excitation, and $\beta > 0$ the strength of lateral inhibition.

Equivalently, the dynamics can be written in matrix-vector form as $\dot{x} + x = [b + Wx]^+$, where $W = \alpha I - \beta J$ includes both self-excitation and lateral inhibition. The state of the network is specified by the vector $x$, and the external input by the vector $b$. A vector $v$ is said to be nonnegative, $v \geq 0$, if all of its components are nonnegative. The nonnegative orthant is the set of all nonnegative vectors. It can be shown that any trajectory of Eq. (4) starting in the nonnegative orthant remains there. Therefore, for simplicity we will consider trajectories that are confined to the nonnegative orthant $x \geq 0$. However, we will consider input vectors $b$ whose components are of arbitrary sign.

# 2 Global stability

The goal of this paper is to characterize the steady state response of the dynamics Eq. (4) to an input $b$ that is constant in time. For this to be a sensible goal, we need some guarantee that the dynamics converges to a steady state, and does not diverge to infinity. This is provided by the following theorem.

**Theorem 1** *Consider the network Eq. (4). The following statements are equivalent:*

1. *For any input b, there is a nonempty set of steady states that is globally asymptotically stable, except for initial conditions in a set of measure zero.*

2. *The strength $\alpha$ of self-excitation is less than one.*

**Proof sketch:**

- $(2) \Rightarrow (1)$: If $\alpha < 1$, the function $\frac{1}{2}(1-\alpha)x^T x + \frac{\beta}{2} x^T J x - b^T x$ is bounded below and radially unbounded in the nonnegative orthant. Furthermore it is nonincreasing under the dynamics Eq. (4), and constant only at steady states. Therefore it is a Lyapunov function, and its local minima are globally asymptotically stable.

- $(1) \Rightarrow (2)$: Suppose that (2) is false. If $\alpha \geq 1$, it is possible to choose $b$ and an initial condition for $x$ so that only one neuron is active, and the activity of this neuron diverges, so that (1) is contradicted. ∎

## 3 Relationship between groups and permitted sets

In this section we characterize the conditions under which the lateral inhibition of Eq. (4) enforces winner-take-all competition between the groups of neurons. That is, the only sets of neurons that can be coactivated at a stable steady state are the groups and their subsets. This is done by performing a linear stability analysis, which allows us to classify active sets using the following definition.

**Definition 1** *If a set of neurons can be coactivated by some input at an asymptotically stable steady state, it is called* permitted. *Otherwise, it is* forbidden

Elsewhere we have shown that whether a set is permitted or forbidden depends on the submatrix of synaptic connections between neurons in that set[1]. If the largest eigenvalue of the sub-matrix is less than unity, then the set is permitted. Otherwise, it is forbidden. We have also proved that any superset of a forbidden set is forbidden, while any subset of a permitted set is also permitted.

Our goal in constructing the network (4) is to make the groups and their subsets the only permitted sets of the network. To determine whether this is the case, we must answer two questions. First, are all groups and their subsets permitted? Second, are all permitted sets contained in groups? The first question is answered by the following Lemma.

**Lemma 1** *All groups and their subsets are permitted.*

**Proof:** If a set is contained in a group, then there is no lateral inhibition between the neurons in the set. Provided that $\alpha < 1$, all eigenvalues of the sub-matrix are less than unity, and the set is permitted. ∎

The answer to the second question, whether all permitted sets are contained in groups, is not necessarily affirmative. For example, consider the network defined by the group membership matrix $\xi = \{(1,1,0), (0,1,1), (1,0,1)\}$. Since every pair of neurons belongs to some group, there is no lateral inhibition ($J = 0$), which means that there are no forbidden sets. As a result, $(1,1,1)$ is a permitted set, but obviously it is not contained in any group.

Let's define a *spurious* permitted set to be one that is not contained in any group. For example, $\{1,1,1\}$ is a spurious permitted set in the above example. To eliminate all the spurious permitted sets in the network, certain conditions on the group membership matrix $\xi$ have to be satisfied.

**Definition 2** *The membership $\xi$ is* degenerate *if there exists a set of $n \geq 3$ neurons that is not contained in any group, but all of its subsets with $n-1$ neurons belong to some group.*

*Otherwise, $\xi$ is called* nondegenerate. *For example, $\xi = \{(1, 1, 0), (0, 1, 1), (1, 0, 1)\}$ is degenerate.*

Using this definition, we can formulate the following theorem.

**Theorem 2** *The neural dynamics Eq. (4) with $\alpha < 1$ and $\beta > 1 - \alpha$ has a spurious permitted set if and only if $\xi$ is degenerate.*

Before we prove this theorem, we will need the following lemma.

**Lemma 2** *If $\beta > 1 - \alpha$, any set containing two neurons not in the same group is forbidden under the neural dynamics Eq. (4).*

**Proof sketch:** We will start by analyzing a very simple case, where there are two neurons belonging to two different groups. Let the group membership be $\{(1, 0), (0, 1)\}$. In this case, $W = \{(\alpha, -\beta), (-\beta, \alpha)\}$. This matrix has eigenvectors $(1, 1)$ and $(1, -1)$ and eigenvalues $\alpha - \beta$ and $\alpha + \beta$. Since $\alpha < 1$ for global stability and $\beta > 0$ by definition, the $(1, 1)$ mode is always stable. But if $\beta > 1 - \alpha$, the $(1, -1)$ mode is unstable. This means that it is impossible for the two neurons to be coactivated at a stable steady state. Since any superset of a forbidden set is also forbidden, the general result of the lemma follows. ∎.

**Proof of Theorem 2 (sketch):**

- $\Leftarrow$: If $\xi$ is degenerate, there must exist a set $n \geq 3$ neurons that is not contained in any group, but all of its subsets with $n - 1$ neurons belong to some group. There is no lateral inhibition between these $n$ neurons, since every pair of neurons belongs to some group. Thus the set containing all $n$ neurons is permitted and spurious.

- $\Rightarrow$: If there exists a spurious permitted set $P$, we need to prove that $\xi$ must be degenerate. We will prove this by contradiction and induction. Let's assume $\xi$ is nondegenerate.

  $P$ must contain at least 2 neurons since any one neuron subset is permitted and not spurious. By Lemma 2, these 2 neurons must be contained in some group, or else it is forbidden. Thus $P$ must contain at least 3 neurons to be spurious, and any pair of neurons in $P$ belongs to some group by Lemma 2.

  If $P$ contains at least $n$ neurons and all of its subsets with $n - 1$ neurons belong to some group, then the set with these $n$ neurons must belong to some group, otherwise $\xi$ is degenerate. Thus $n$ must contain at least $n + 1$ neurons to be spurious, and all its $n$ subsets belong to some group.

  By induction, this implies that $P$ must contain all neurons in the network, in which case, $P$ is either forbidden or nonspurious. This contradicts with the assumption $P$ is a spurious permitted set. ∎

From Theorem 2, we can easily have the following result.

**Corollary 1** *If every group contains some neuron that does not belong to any other group, then there is no any spurious permitted set.*

## 4 The potential winners

We have seen that if $\xi$ is nondegenerate, the active set must be contained in a group, provided that lateral inhibition is strong ($\beta > 1 - \alpha$). The group that contains the active set will be called the "winner" of the competition between groups. The identity of the winner depends on the input $b$, and also on the initial conditions of the dynamics. For a given input, we need to characterize which pattern could potentially be the winner.

Suppose that the *group inputs* $B^a = \sum_i [b_i]^+ \xi_i^a$ are distinct. Without loss of generality, we order the group inputs as $B^1 > \ldots > B^m$. Let's denote the largest input as $b_{max} = \max_i \{b_i\}$ and assume $b_{max} > 0$.

**Theorem 3** *For nonoverlapping groups, the top c groups with the largest group input could end up the winner depending on the initial conditions of the dynamics, where c is determined by the equation* $B^c \geq (1 - \alpha)\beta^{-1} b_{max} > B^{c+1}$

**Proof sketch:** Suppose the $a$th group is the winner. For all neurons not in this group to be inactive, the self-consistent condition should read

$$\sum_i [b_i]^+ \xi_i^a \geq \frac{1 - \alpha}{\beta} \max_{j \notin a}\{[b_j]^+\} \tag{5}$$

If a group containing the neuron with the largest input, this condition can always be satisfied. Moreover, this group is always in the top $c$ groups. For groups not containing the neuron with the largest input, this condition can be satisfied if and only if they are in the top $c$ groups.∎

The winner-take-all competition described above holds only for the case of strong inhibition $\beta > 1 - \alpha$. On the other hand, if $\beta$ is small, the competition will be weak and may not result in group-winner-take-all. In particular, if $\beta < (1 - \alpha)/\lambda_{max}$, where $\lambda_{max}$ is the largest eigenvalue of $-J$, then the set of all neurons is permitted. Since every subset of a permitted set is permitted, that means there are no forbidden sets and the network is monostable. Hence, group-winner-take-all does not hold. If $(1 - \alpha)/\lambda_{max} < \beta < 1 - \alpha$, the network has forbidden sets, but the possibility of spurious permitted sets cannot be excluded.

# 5 Examples

**Traditional winner-take-all network** This is a special case of our network with $N$ groups, each containing one of the $N$ neurons. Therefore, the group membership matrix $\xi$ is the identity matrix, and $J = 11^T - I$, where 1 denotes the vector of all ones. According to Corollary 1, only one neuron is permitted to be active at a stable steady state, provided that $\beta > 1 - \alpha$. We refer to the active neuron as the "winner" of the competition mediated by the lateral inhibition.

If we assume that the inputs $b_i$ have distinct values, they can be ordered as $b_1 > b_2 > \cdots > b_N$, without loss of generality. According to Theorem 3, any of the neurons 1 to $k$ can be the winner, where $k$ is defined by $b_k \geq (1 - \alpha)\beta^{-1} b_1 > b_{k+1}$. The winner depends on the initial condition of the network dynamics. In other words, any neuron whose input is greater than $(1 - \alpha)/\beta$ times the largest input can end up the winner.

**Topographic organization** Let the $N$ neurons be organized into a ring, and let every set of $d$ contiguous neurons be a group. $d$ will be called the width. For example, in a network with $N = 4$ neurons and group width $d = 2$, then the membership matrix is $\xi = \{(1,1,0,0), (0,1,1,0), (0,0,1,1), (1,0,0,1)\}$. This ring network is similar to the one proposed by Ben-Yishai et al in the modeling of orientation tuning of visual cortex[9].

Unlike the WTA network where all groups are non-overlapping which implies that $\xi$ is always nondegenerate, in the ring network neurons are shared among different groups, $\xi$ will become degenerate when the width of the group is large. To guarantee all permitted sets are the subsets of some group, we have the following corollary, which can be derived from Theorem 2.

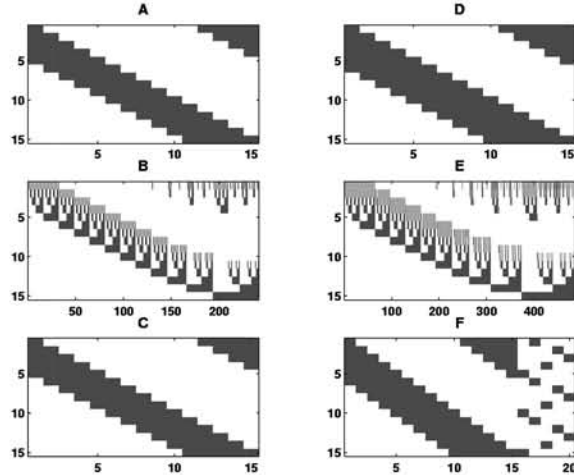

Figure 1: Permitted sets of the ring network. The ring network is comprised of 15 neurons with $\alpha = 0.4$ and $\beta = 1$. In panels A and D, the 15 groups are represented by columns. Black refers to active neurons and white refers to inactive neurons. (A) 15 groups of width $d = 5$. (B) All permitted sets corresponding to the groups in A. (C) The 15 permitted sets in B that have no permitted supersets. They are the same as the groups in A. (D) 15 groups with width $d = 6$. (E) All permitted set corresponding to groups in D. (F) There are 20 permitted sets in E that have no permitted supersets. Note that there are 5 spurious permitted sets.

**Corollary 2** *In the ring network with $N$ neurons, if the width $d < N/3 + 1$, then there is no spurious permitted set.*

Fig. (1) shows the permitted sets of a ring network with 15 neurons. From Corollary 2, we know that if the group width is no larger than 5 neurons, there will not exist any spurious permitted set. In the left three panels of Fig. (1), the group width is 5 and all permitted sets are subsets of these groups. However, when the group width is 6 (right three panels), there exists 5 spurious permitted sets as shown in panel F.

As we have mentioned earlier, the lateral inhibition strength $\beta$ plays a critical role in determining the dynamics of the network. Fig. (2) shows four types of steady states of a ring network corresponding to different values of $\beta$.

## 6 Discussion

We have shown that it is possible to organize lateral inhibition to mediate a winner-take-all competition between potentially overlapping groups of neurons. Our construction utilizes the distinction between permitted and forbidden sets of neurons.

If there is strong lateral inhibition between two neurons, then any set that contains them is forbidden (Lemma 2). Neurons that belong to the same group do not have any mutual inhibition, and so they form a permitted set. Because the synaptic connections between neurons in the same group are only composed of self-excitation, their outputs equal their rectified inputs, amplified by the gain factor of $1/(1 - \alpha)$. Hence the neurons in the winning group operate in a purely analog regime. The coexistence of analog filtering with logical constraints on neural activation represents a form of hybrid analog-digital computation that may be especially appropriate for perceptual tasks. It might be possible to apply a similar method to the problem of data re-

construction using a constrained set of basis vectors. The constraints on the linear combination of basis vectors could for example implement sparsity or nonnegativity constraints.

As we have shown in Theorem 2, there are some degenerate cases of overlapping groups, to which our method does not apply. It is an interesting open question whether there exists a general way of how to translate arbitrary groups of coactive neurons into permitted sets without involving spurious permitted sets.

In the past, a great deal of research has been inspired by the idea of storing memories as dynamical attractors in neural networks [10]. Our theory suggests an alternative viewpoint, which is to regard permitted sets as memories latent in the synaptic connections. From this viewpoint, the contribution of the present paper is a method of storing and retrieving memories as permitted sets in neural networks.

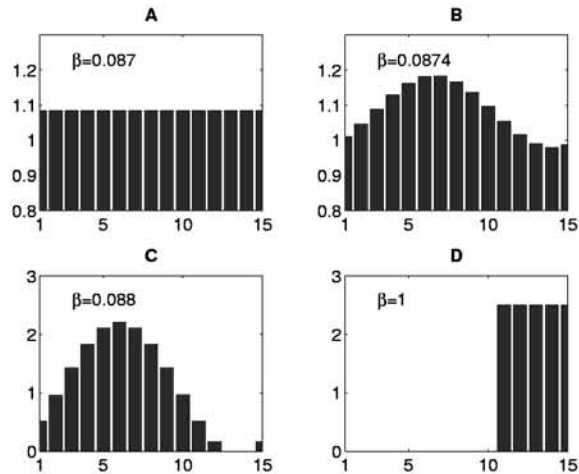

Figure 2: Lateral inhibition strength $\beta$ determines the behavior of the network. The network is a ring network of 15 neurons with width $d = 5$ and where $\alpha = 0.4$ and input $b_i = 1, \forall i$. These panels show the steady state activities of the 15 neurons. (A) There are no forbidden sets. (B) The marginal state $\beta = (1 - \alpha)/\lambda_{max} = 0.874$, in which the network forms a continuous attractor. (C) Forbidden sets exist, and so do spurious permitted sets. (D) Group-winner-take-all case, no spurious permitted sets.

## Footnotes

[1]This condition can be relaxed, but is kept for simplicity.

# References

[1] R. Hahnloser, R. Sarpeshkar, M. Mahowald, Douglas R., and H.S. Seung. Digital selection and analog amplification coexist in an electronic circuit inspired by neocortex. *Nature*, 3:609–616, 2000.

[2] Shun-Ichi Amari and Michael A. Arbib. *Competition and Cooperation in Neural Nets*, pages 119–165. Systems Neuroscience. Academic Press, 1977. J. Metzler (ed).

[3] J. Feng and K.P. Hadeler. Qualitative behaviour of some simple networks. *J. Phys. A:*, 29:5019–5033, 1996.

[4] Richard H.R. Hahnloser. About the piecewise analysis of networks of linear threshold neurons. *Neural Networks*, 11:691–697, 1998.

[5] T. Kohonen. *Self-Organization and Associative Memory*. Springer-Verlag, Berlin, 3 edition, 1989.

[6] D. D. Lee and H. S. Seung. Learning the parts of objects by nonnegative matrix factorization. *Nature*, 401:788–91, 1999.

[7] B. A. Olshausen and D. J. Field. Emergence of simple-cell receptive field properties by learning a sparse code for natural images. *Nature*, 381:607–609, 1996.

[8] R. Ben-Yishai, R. Lev Bar-Or, and H. Sompolinsky. Theory of orientation tuning in visual cortex. *Proc. Natl. Acad. Sci. USA*, 92:3844–3848, 1995.

[9] J. J. Hopfield. Neurons with graded response have collective properties like those of two-state neurons. *Proc. Natl. Acad. Sci. USA*, 81:3088–3092, 1984.
